# Blending Autonomous Exploration and Apprenticeship Learning

**Thomas J. Walsh**
Center for Educational
Testing and Evaluation
University of Kansas
Lawrence, KS 66045
twalsh@ku.edu

**Daniel Hewlett**    **Clayton T. Morrison**
School of Information:
Science, Technology and Arts
University of Arizona
Tucson, AZ 85721
{dhewlett@cs,clayton@sista}.arizona.edu

## Abstract

We present theoretical and empirical results for a framework that combines the benefits of apprenticeship and autonomous reinforcement learning. Our approach modifies an existing apprenticeship learning framework that relies on teacher demonstrations and does not necessarily explore the environment. The first change is replacing previously used Mistake Bound model learners with a recently proposed framework that melds the KWIK and Mistake Bound supervised learning protocols. The second change is introducing a communication of expected utility from the student to the teacher. The resulting system only uses teacher traces when the agent needs to learn concepts it cannot efficiently learn on its own.

## 1   Introduction

As problem domains become more complex, human guidance becomes increasingly necessary to improve agent performance. For instance, *apprenticeship learning*, where teachers demonstrate behaviors for agents to follow, has been used to train agents to control complicated systems such as helicopters [1]. However, most work on this topic burdens the teacher with demonstrating even the simplest nuances of a task. By contrast, in *autonomous* reinforcement learning [2] a number of domain classes can be efficiently learned by an actively exploring agent, although this class is provably smaller than those learnable with the help of a teacher [3].

Thus the field seems to be largely bifurcated. Either agents learn autonomously and eschew the larger learning capacity from teacher interaction, or the agent overburdens the teacher by not exploring simple concepts it could garner on its own. Intuitively, this seems like a false choice, as human teachers often use demonstration but also let students explore parts of the domain on their own. We show how to build a provably efficient learning system that balances teacher demonstrations and autonomous exploration. Specifically, our protocol and algorithms cause a teacher to only step in when its advice will be significantly more helpful than autonomous exploration by the agent.

We extend a previously proposed apprenticeship learning protocol [3] where a learning agent and teacher take turns running trajectories. This version of apprenticeship learning is fundamentally different from Inverse Reinforcement Learning [4] and imitation learning [5] because our agents are allowed to enact better policies than their teachers and observe reward signals. In this setting, the number of times the teacher outperforms the student was proven to be related to the learnability of the domain class in a *mistake bound predictor* (MBP) framework.

Our work modifies previous apprenticeship learning efforts in two ways. First, we will show that replacing the MBP framework with a different learning architecture called KWIK-MBP (based on a similar recently proposed protocol [6]) indicates areas where the agent should autonomously explore, and melds autonomous and apprenticeship learning. However, this change alone is not suffi-

cient to keep the teacher from intervening when an agent is capable of learning on its own. Hence, we introduce a communication of the agent's expected utility, which provides enough information for the teacher to decide whether or not to provide a trace (a property not shared by any of the previous efforts). Furthermore, we show the number of such interactions grows only with the MBP portion of the KWIK-MBP bound. We then discuss how to relax the communication requirement when the teacher observes the student for many episodes. This gives us the first apprenticeship learning framework where a teacher only shows demonstrations when they are needed for efficient learning, and gracefully blends autonomous exploration and apprenticeship learning.

## 2 Background

The main focus of this paper is blending KWIK autonomous exploration strategies [7] and apprenticeship learning techniques [3], utilizing a framework for measuring mistakes and uncertainty based on KWIK-MB [6]. We begin by reviewing results relating the learnability of domain parameters in a supervised setting to the efficiency of model-based RL agents.

### 2.1 MDPs and KWIK Autonomous Learning

We will consider environments modeled as a Markov Decision Process (MDP) [2] $\langle S, A, T, R, \gamma \rangle$, with states and actions $S$ and $A$, transition function $T : S, A \mapsto Pr[S]$, rewards $R : S, A \mapsto \Re$, and discount factor $\gamma \in [0, 1)$. The *value* of a state under policy $\pi : S \mapsto A$ is $V_\pi(s) = R(s, \pi(s)) + \gamma \sum_{s' \in S} T(s, a, s') V_\pi(s')$ and the optimal policy $\pi^*$ satisfies $\forall_\pi V_{\pi*} \geq V_\pi$.

In model-based reinforcement learning, recent advancements [7] have linked the efficient learnability of $T$ and $R$ in the KWIK ("Knows What It Knows") framework for supervised learning with PAC-MDP behavior [8]. Formally, KWIK learning is:

**Definition 1.** *A hypothesis class $H : X \mapsto Y$ is* KWIK *learnable* with parameters $\epsilon$ and $\delta$ if the following holds. For each (adversarial) input $x_t$ the learner predicts $y_t \in Y$ or "I don't know" ($\perp$). With probability $(1 - \delta)$ (1) when $y_t \neq \perp$, $||y_t - E[h(x_t)]|| < \epsilon$ and (2) the total number of $\perp$ predictions is bounded by a polynomial function of $(|H|, \frac{1}{\epsilon}, \frac{1}{\delta})$.*

Intuitively, KWIK caps the number of times the agent will admit uncertainty in its predictions. Prior work [7] showed that if the transition and reward functions ($T$ and $R$) of an MDP are KWIK learnable, then a PAC-MDP agent (which takes only a polynomial number of suboptimal steps with high probability) can be constructed for autonomous exploration. The mechanism for this construction is an *optimistic interpretation* of the learned model. Specifically, KWIK-learners $L_T$ and $L_R$ are built for $T$ and $R$ and the agent replaces any $\perp$ predictions with transitions to a trap state with reward $R_{max}$, causing the agent to explore these uncertain regions. This exploration requires only a polynomial (with respect to the domain parameters) number of suboptimal steps, thus the link from KWIK to PAC-MDP. While the class of functions that is KWIK learnable includes tabular and factored MDPs, it does not cover many larger dynamics classes (such as STRIPS rules with conjunctions for pre-conditions) that are efficiently learnable in the apprenticeship setting.

### 2.2 Apprenticeship Learning with Mistake Bound Predictor

We now describe an existing apprenticeship learning framework [3], which we will be modifying throughout this paper. In that protocol, an agent is presented with a start state $s_0$ and is asked to take actions according to its current policy $\pi_A$, until a horizon $H$ or a terminal state is reached. After each of these episodes, a teacher is allowed to (but may choose not to) demonstrate their own policy $\pi_T$ starting from $s_0$. The learning agent is able to fully observe each transition and reward received both in its own trajectories as well as those of the teacher, who may be able to provide highly informative samples. For example, in an environment with $n$ bits representing a *combination lock* that can only be opened with a single setting of the bits, the teacher can demonstrate the combination in a single trace, while an autonomous agent could spend $2^n$ steps trying to open it.

Also in that work, the authors describe a measure of sample complexity called *PAC-MDP-Trace* (analogous to PAC-MDP from above) that measures (with probability $1 - \delta$) the number of episodes where $V_{\pi_A}(s_0) < V_{\pi_T}(s_0) - \epsilon$, that is where the expected value of the agent's policy is significantly worse than the expected value of the teacher's policy ($V_A$ and $V_T$ for short). A result analogous

to the KWIK to PAC-MDP result was shown connecting a supervised framework called *Mistake Bound Predictor* (MBP) to PAC-MDP-Trace behavior. MBP extends the classic mistake bound learning framework [9] to handle data with noisy labels, or more specifically:

**Definition 2.** *A hypothesis class $H : X \mapsto Y$ is Mistake Bound Predictor (MBP) learnable with parameters $\epsilon$ and $\delta$ if the following holds. For each adversarial input $x_t$, the learner predicts $y_t \in Y$. If $||E_{h*}[x_t] - y_t|| > \epsilon$, then the agent has made a* mistake. *The number of mistakes must be bounded by a polynomial over $(\frac{1}{\epsilon}, \frac{1}{\delta}, |H|)$ with probability $(1 - \delta)$.*

An agent using MBP learners $L_T$ and $L_R$ for the MDP model components will be PAC-MDP-Trace. The conversion mirrors the KWIK to PAC-MDP connection described earlier, except that the interpretation of the model is strict, and often *pessimistic* (sometimes resulting in an underestimate of the value function). For instance, if the transition function is based on a conjunction (e.g. our combination lock), the MBP learners default to predicting "false" where the data is incomplete, leading an agent to think its action will not work in those situations. Such interpretations would be catastrophic in the autonomous case (where the agent would fail to explore such areas), but are permissible in apprenticeship learning where teacher traces will provide the missing data.

Notice that under a criteria where the number of teacher traces is to be minimized, MBP learning may overburden the teacher. For example, in a simple flat MDP, an MBP-Agent picks actions that maximize utility in the part of the state space that has been exposed by the teacher, never exploring, so the number of teacher traces scales with $|S||A|$. But a flat MDP is autonomously (KWIK) learnable, so no traces should be required. Ideally an agent would explore the state space where it can learn efficiently, and only rely on the teacher for difficult to learn concepts (like conjunctions).

## 3   Teaching by Demonstration with Mixed Interpretations

We now introduce a different criteria with the goal of minimizing teacher traces while not forcing the agent to explore exponentially long.

**Definition 3.** *A* Teacher Interaction *(TI) bound for a student-teacher pair is the number of episodes where the teacher provides a trace to the agent that guarantees (with probability $1 - \delta$) that the number of agent steps between each trace (or after the last one) where $V_A(s_0) < V_T(s_0) - \epsilon$ is polynomial in $\frac{1}{\epsilon}, \frac{1}{\delta}$, and the domain parameters.*

A good TI bound minimizes the teacher traces needed to achieve good behavior, but only requires the suboptimal exploration steps to be polynomially bounded, not minimized. This reflects our judgement that teacher interactions are far more costly than autonomous agent steps, so as long as the latter are reasonably constrained, we should seek to minimize the former. The relationship between TI and PAC-MDP-Trace is the following:

**Theorem 1.** *The TI bound for a domain class and learning algorithm is upper-bounded by the PAC-MDP-Trace bound for the same domain/algorithm with the same $\epsilon$ and $\delta$ parameters.*

*Proof.* A PAC-MDP-Trace bound quantifies (with probability $1 - \delta$) the worst-case number of episodes where the student performs worse than the teacher, specifically where $V_A(s_0) < V_T(s_0) - \epsilon$. Suppose an environment existed with a PAC-MDP-Trace bound of $B_1$ and a TI of $B_2 > B_1$. This would mean the domain was learnable with at most $B_1$ teacher traces. But this is a contradiction because no more traces are needed to keep the autonomous exploration steps polynomial. $\square$

### 3.1   The KWIK-MBP Protocol

We would like to describe a supervised learning framework (like KWIK or MBP) that can quantify the number of changes made to a model through exploration and teacher demonstrations. Here, we propose such a model based on the recent KWIK-MB protocol [6], which we extend below to cover stochastic labels (KWIK-MBP).

**Definition 4.** *A hypothesis class $H : X \mapsto Y$ is KWIK-MBP with parameters $\epsilon$ and $\delta$ under the following conditions. For each (adversarial) input $x_t$ the learner must predict $y_t \in Y$ or $\perp$. With probability $(1 - \delta)$, the number of $\perp$ predictions must be bounded by a polynomial $K$ over $\langle |H|, 1/\epsilon, 1/\delta \rangle$ and the number of mistakes (by Definition 2) must be bounded by a polynomial $M$ over $\langle |H|, 1/\epsilon, 1/\delta \rangle$.*

---
**Algorithm 1** KWIK-MBP-Agent with Value Communication
---
1: The agent $\mathcal{A}$ knows $\epsilon$, $\delta$, $S$, $A$, $H$ and planner $P$.
2: The teacher $\mathcal{T}$ has policy $\pi_T$ with expected value $V_T$
3: Initialize KWIK-MBP learners $L_T$ and $L_R$ to ensure $\frac{\epsilon}{k}$ value accuracy w.h.p. for $k \geq 2$
4: **for** each episode **do**
5:     $s_0 = Environment.startState$
6:     $\mathcal{A}$ calculates the value function $U_A$ of $\pi_A$ from $\hat{S}$, $A$, $\hat{T}$ and $\hat{R}$ (see construction below).
7:     $\mathcal{A}$ communicates its expected utility $U_A(s_0)$ on this episode to $\mathcal{T}$
8:     **if** $V_T(s_0) - \frac{k-1}{k}\epsilon > U_A(s_0)$ **then**
9:         $\mathcal{T}$ provides a trace $\tau$ starting from $s_0$.
10:         $\forall \langle s, a, r, s' \rangle$ Update $L_T(s, a, s')$ and $L_R(s, a, r)$
11:     **while** episode not finished and $t < H$ **do**
12:         $\hat{S} = S \bigcup S_{max}$, the $R_{max}$ trap state
13:         $\hat{R} = L_R(s, a)$ or $R_{max}$ if $L_R(s, a) = \bot$
14:         $\hat{T} = L_T(s, a)$ or $S_{max}$ if $L_T(s, a) = \bot$.
15:         $a_t = P.getPlan(s_t, \hat{S}, \hat{T}, \hat{R})$.
16:         $\langle r_t, s_{t+1} \rangle = E.executeAct(a_t)$
17:         $L_T.Update(s_t, a_t, s_{t+1}); L_R.Update(s_t, a_t, r_t)$
---

KWIK-MB was originally designed for a situation where mistakes are more costly than $\bot$ predictions. So mistakes are minimized while $\bot$ predictions are only bounded. This is analogous to our TI criteria (traces minimized with exploration bounded) so we now examine a KWIK-MBP learner in the apprenticeship setting.

## 3.2 Mixing Optimism and Pessimism

Algorithm 1 (KWIK-MBP-Agent) shows an apprenticeship learning agent built over KWIK-MBP learners $L_T$ and $L_R$. Both of these model learners are instantiated to ensure the learned value function will have $\frac{\epsilon}{k}$ accuracy for $k \geq 2$ (for reasons discussed in the main theorem), which can be done by setting $\epsilon_R = \epsilon \frac{1-\gamma}{16k}$ and $\epsilon_T = \epsilon \frac{(1-\gamma)^2}{16k\gamma V_{max}}$ (details follow the same form as standard connections between model learners and value function accuracy, for example in Theorem 3 from [7]). When planning with the subsequent model, the agent constructs a "mixed" interpretation, trusting the learner's predictions where mistakes might be made, but replacing (lines 13-14) all $\bot$ predictions from $L_R$ with a reward of $R_{max}$ and any $\bot$ predictions from $\hat{T}$ with transitions to the $R_{max}$ trap state $S_{max}$. This has the effect of drawing the agent to explore explicitly uncertain regions ($\bot$) and to either explore on its own or rely on the teacher for areas where a mistake might be made. For instance, in the experiments in Figure 2 (left), discussed in depth later, a KWIK-MBP agent only requires traces for learning the pre-conditions in a noisy blocks world but uses autonomous exploration to discover the noise probabilities.

## 4 Teaching by Demonstration with Explicit Communication

Thus far we have not discussed communication from the student to the teacher in KWIK-MBP-Agent (line 7). We now show that this communication is vital in keeping the TI bound small.

**Example 1.** *Suppose there was no communication in Algorithm 1 and the teacher provided a trace when $\pi_A$ was suboptimal. Consider a domain where the pre-conditions of actions are governed by a disjunction over the $n$ state factors (if the disjunction fails, the action fails). Disjunctions can be learned with $M = n/3$ mistakes and $K = 3n/2 - 3M$ $\bot$ predictions [6]. However, that algorithm defaults to predicting "true" and only learns from negative examples. This optimistic interpretation means the agent will expect success, and can learn autonomously. However, the teacher will provide a trace to the agent since it sees it performing suboptimally during exploration. Such traces are unnecessary and uninformative (their positive examples are useless to $L_T$).*

This illustrates the need for student communication to give some indication of its internal model to the teacher. The protocol in Algorithm 1 captures this intuition by providing a channel (line 7)

where the student communicates its expected utility $U_A$. The teacher then only shows a trace to a pessimistic agent (line 8), but will "stand back" and let an over-confident student learn from its own mistakes. We note that there are many other possible forms of this communication such as announcing the probability of reaching a goal or an equivalence query [10] type model, where the student exposes its entire internal model to the teacher. We focus here on the communication of utility, which is general enough for MDP domains but has low communication overhead.

## 4.1 Theoretical Properties

The proof of the algorithm's TI bound appears below and is illustrated in Figure 1 but intuitively we show that if we force the student to (w.h.p.) learn an $\frac{\epsilon}{k}$-accurate value function for $k \geq 2$ then we can guarantee traces where $\overline{U_A} < V_T - \frac{\epsilon}{k}$ will be helpful, but are not needed until $U_A$ is reported below $V_T - \frac{(k-1)}{k}\epsilon$, at which point $U_A$ alone cannot guarantee that $V_A$ is within $\epsilon$ of $V_T$ and so a trace must be given. Because traces are only given when the student undervalues a potential policy, the number of traces is related only to the

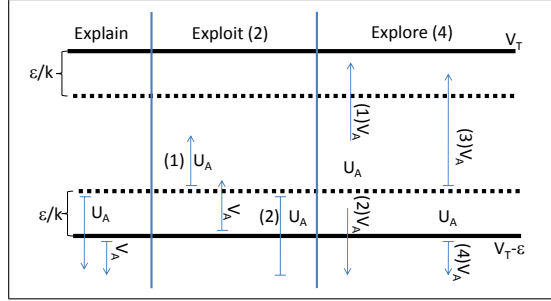

Figure 1: The areas for $U_A$ and $V_A$ corresponding to the cases in the main theorem. In all cases $U_T \leq U_A$ and when $k = 2$ the two dashed lines collapse together.

MBP portion of the KWIK-MBP bound, and more specifically to the number of pessimistic mistakes, defined as:

**Definition 5.** *A mistake is* pessimistic *if and only if it causes some policy $\pi$ to be undervalued in the agent's model, that is in our case $U_\pi < V_\pi - \frac{\epsilon}{k}$.*

Note that by the construction of our model, KWIK-learnable parameters ($\perp$ replaced by Rmax-style interpretations) never result in such pessimistic mistakes. We can now state the following:

**Theorem 2.** *Algorithm 1 with KWIK-MBP learners will have a TI bound that is polynomial in $\frac{1}{\epsilon}$, $\frac{1}{\delta}$, and $\frac{1}{1-\gamma}$ and $P$, where $P$ is the number of pessimistic mistakes ($P \leq M$) made by $L_T$ and $L_R$.*

*Proof.* The proof stems from an expansion of the Explore-Explain-Exploit Lemma from [3]. That original lemma categorized the three possible outcomes of an episode in an apprenticeship learning setting where the teacher *always* gives a trace and with $L_T$ and $L_R$ built to learn $V$ within $\frac{\epsilon}{2}$. The three possibilities for an episode were (1) *exploration*, when the agent's value estimate of $\pi_A$ is inaccurate, $||V_A - U_A|| > \epsilon/2$, (2) *exploitation* when the agent's prediction of its own return is accurate ($||U_A - V_A|| \leq \epsilon/2$) and the agent is near-optimal with respect to the teacher ($V_A \geq V_T - \epsilon$), and (3) *explanation* when $||V_A - U_A|| \leq \epsilon/2$, but $V_A < V_T - \epsilon$. Because both (1) and (3) provide samples to $L_T$ and $L_R$, the number of times they can occur is bounded (in the original lemma) by the MBP bound on those learners and in both cases a relevant sample is produced with high probability due to the simulation lemma (c.f. Lemma 9 of [7]), which states that two different value returns from two MDPs means that, with high probability, their parameters must be different.

We need to extend the lemma to cover our change in protocol (the teacher may not step in on every episode) and in evaluation criteria (TI bound instead of PAC-MDP-Trace). Specifically, we need to show: **(i)** The number of steps between traces where $V_A < V_T - \epsilon$ is polynomially bounded. **(ii)** Only a polynomial number of traces are given, and they are all guaranteed to improve some parameter in the agent's model with high probability. **(iii)** Only pessimistic mistakes (Definition 5) cause a teacher intervention. Note that properties (i) and (ii) imply that $V_A < V_T - \epsilon$ for only a polynomial number of episodes and correspond directly to the TI criteria from Definition 3. We now consider Algorithm 1 according to these properties in all of the cases from the original explore-exploit-explain lemma.

We begin with the Explain case where $V_A < V_T - \epsilon$ and $||U_A - V_A|| \leq \frac{\epsilon}{k}$. Combining these inequalities, we know $U_A < V_T - \frac{\epsilon(k-1)}{k}$, so a trace will definitely be provided. Since $U_T \leq U_A$ ($U_T$ is the value of $\pi_T$ in the student's model and $U_A$ was optimal) we have at least $U_T < V_T - \frac{\epsilon}{k}$ and the simulation lemma implies the trace will (with high probability) be helpful. Since there are a

limited number of such mistakes (because $L_R$ and $L_T$ are KWIK-MBP learners) we have satisfied property (ii). Property (iii) is true because both $\pi_T$ and $\pi_A$ are undervalued.

We now consider the Exploit case where $V_A \geq V_T - \epsilon$ and $||U_A - V_A|| \leq \frac{\epsilon}{k}$. There are two possible situations here, because $U_A$ can either be larger or smaller than $V_T - \frac{\epsilon(k-1)}{k}$. If $U_A \geq V_T - \frac{\epsilon(k-1)}{k}$ then no trace is given, but the agent's policy is near optimal so property (i) is not violated. If $U_A < V_T - \frac{\epsilon(k-1)}{k}$, then a trace is given, even in this exploit case, because the teacher does not know $V_A$ and cannot distinguish this case from the "explain" case above. However, this trace will still be helpful, because $U_T \leq U_A$, so at least $U_T < V_T - \frac{\epsilon}{k}$ (satisfying iii), and again by the simulation lemma, the trace will help us learn a parameter and there are a limited number of such mistakes, so (ii) holds.

Finally, we have the Explore case, where $||U_A - V_A|| > \frac{\epsilon}{k}$. In that case, the agent's own experience will help it learn a parameter, but in terms of traces we have the following cases:

$U_A \geq V_T - \frac{\epsilon(k-1)}{k}$ and $V_A > U_A + \frac{\epsilon}{k}$. In this case no trace is given but we have $V_A > V_T - \epsilon$, so property (i) holds.

$U_A \geq V_T - \frac{\epsilon(k-1)}{k}$ and $U_A > V_A + \frac{\epsilon}{k}$. No trace is given here, but this is the classical exploration case ($U_A$ is optimistic, as in KWIK learning). Since $U_A$ and $V_A$ are sufficiently separated, the agent's own experience will provide a useful sample, and because all parameters are polynomially learnable, property (i) is satisfied.

$U_A < V_T - \frac{\epsilon(k-1)}{k}$ and either $V_A > U_A + \frac{\epsilon}{k}$ or $U_A > V_A + \frac{\epsilon}{k}$. In either case, a trace will be provided but $U_T \leq U_A$ so at least $U_T < V_T - \frac{\epsilon}{k}$ and the trace will be helpful (satisfying property (ii)). Pessimistic mistakes are causing the trace (property iii) since $\pi_T$ is undervalued.

$\square$

Our result improves on previous results by attempting to minimize the number of traces while reasonably bounding exploration. The result also generalizes earlier apprenticeship learning results on $\frac{\epsilon}{2}$-accurate learners [3] to $\frac{\epsilon}{k}$-accuracy, while ensuring a more practical and stronger bound (TI instead of PAC-MDP-Trace). The choice of $k$ in this situation is somewhat complicated. Larger $k$ requires more accuracy of the learned model, but decreases the size of the "bottom region" above where a limited number of traces may be given to an already near-optimal agent. So increasing $k$ can either increase or decrease the number of traces, depending on the exact problem instance.

## 4.2 Experiments

We now present experiments in two domains. The first domain is a blocks world with dynamics based on *stochastic* STRIPS operators, a $-1$ step cost, and a goal of stacking the blocks. That is, the environment state is described as a set of grounded relations (e.g. *On*(a, b)) and actions are described by relational (with variables) operators that have conjunctive pre-conditions that must hold for the action to execute (e.g. *putDown*(X, To) cannot execute unless the agent is holding X and To is clear and a block). If the pre-conditions hold, then one of a set of possible effects (pairs of Add and Delete lists), chosen based on a probability distribution over effects, will change the current state. The actions in our blocks world are two versions of *pickup*(X, From) and two versions of *putDown*(X, To), with one version being "reliable", producing the expected result 80% of the time and otherwise doing nothing. The other version of each action has the probabilities reversed. The literals in the effects of the STRIPS operators (the Add and Delete lists) are given to the learning agents, but the *pre-conditions* and the *probabilities* of the effects need to be learned. This is an interesting case because the effect probabilities can be learned autonomously while the conjunctive pre-conditions (of sizes 3 and 4), require teacher input (like our combination lock example).

Figure 2, column **1**, shows KWIK, MBP, and KWIK-MBP agents as trained by a teacher who uses *un*reliable actions half the time. The KWIK learner never receives traces (since its expected utility, shown in **1a**, is always high), but spends an exponential (in the number of literals) time exploring the potential pre-conditions of actions (**1b**). In contrast, the MBP and KWIK-MBP agents use the first trace to learn the pre-conditions. The proportion of trials (out of 30) that the MBP and KWIK-MBP learners received teacher traces across episodes is shown in the bar graphs **1c** and **1d** of Fig. 2. The MBP learner continues to get traces for several episodes afterwards, using them to

help learn the probabilities well after the pre-conditions are learned. This probability learning could be accomplished autonomously, but the MBP pessimistic value function prevents such exploration in this case. By contrast, KWIK-MBP receives 1 trace to learn the pre-conditions, and then explores the probabilities on its own. KWIK-MBP actually learns the probabilities faster than MBP because it targets areas it does not know about rather than relying on potentially redundant teacher samples. However, in rare cases KWIK-MBP receives additional traces; in fact there were two exceptions in the 30 trials, indicated by $*$'s at episodes 5 and 19 in **1d**. The reason for this is that sometimes the learner may be unlucky and construct an inaccurate value estimate and the teacher then steps in and provides a trace.

The second domain is a variant of "Wumpus World" with 5 locations in a chain, an agent who can move, fire arrows (unlimited supply) or pick berries (also unlimited), and a wumpus moving randomly. The domain is represented by a Dynamic Bayes Net (DBN) based on these factors and the reward is represented as a linear combination of the factor values ($-5$ for a live wumpus and $+2$ for picking a berry). The action effects are noisy, especially the probability of killing the wumpus, which depends on the exact (not just relative) locations of the agent, wumpus, and whether the wumpus is dead yet (three parent factors in the DBN). While the reward function is KWIK learnable through linear regression [7] and though DBN CPTs with small parent sizes are also KWIK learnable, the high connectivity of this particular DBN makes autonomous exploration of all the parent-value configurations prohibitive. Because of this, in our KWIK-MBP implementation, we combined a KWIK linear regression learner for $L_R$ with an MBP learner for $L_T$ that is given the DBN structure and learns the parameters from experience, but when entries in the conditional probability tables are the result of

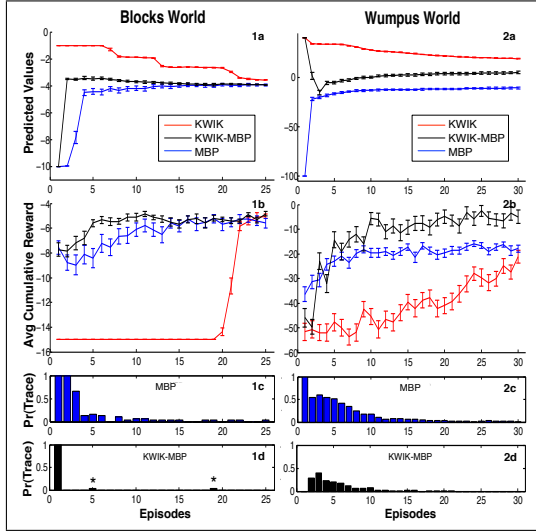

Figure 2: A plot matrix with rows (**a**) value predictions $U_A(s_0)$, (**b**) average undiscounted cumulative reward and (**c** and **d**) the proportion of trials where MBP and KWIK-MBP received teacher traces. The left column is Blocks World and the right a modified Wumpus World. Red corresponds to KWIK, blue to MBP, and black to KWIK-MBP.

only a few data points, the learner predicts no change for this factor, which was generally a pessimistic outcome. We constructed an "optimal hunting" teacher that finds the best combination of locations to shoot the wumpus from/at, but ignores the berries. We concentrate on the ability of our algorithm to find a better policy than the teacher (i.e., learning to pick berries), while staying close enough to the teacher's traces that it can hunt the wumpus effectively.

Figure 2, column **2**, presents the results from this experiment. In plot **2a** we see the predicted values of the three learners, while plot **2b** shows their performance. The KWIK learner starts with high $U_A$ that gradually descends (in **2a**), but without traces the agent spends most of its time exploring fruitlessly (very slowly inclining slope of **2b**). The MBP agent learns to hunt from the teacher and quickly achieves good behavior, but rarely learns to pick berries (only gaining experience on the reward of berries if it ends up in completely unknown state and picks berries at random many times). The KWIK-MBP learner starts with high expected utility and explores the structure of just the reward function, discovering berries but not the proper location combinations for killing the wumpus. Its $U_A$ thus initially drops precipitously as it thinks all it can do is collect berries. Once this crosses the teacher's threshold, the teacher steps in with a number of traces showing the best way to hunt the wumpus—this is seen in plot **2d** with the small bump in the proportion of trials with traces, starting at episode 2 and declining roughly linearly until episode 10. The KWIK-MBP student is then able to fill in the CPTs with information from the teacher and reach an optimal policy that kills the wumpus and picks berries, avoiding both the over- and under-exploration of the KWIK and MBP agents. This increased overall performance is seen in plot **2b** as KWIK-MBP's average cumulative reward surpasses MBP between episodes 5 and 10 .

# 5 Inferring Student Aptitude

We now describe a method for a teacher to infer the student's aptitude by using long periods without teacher interventions as *observation phases*. This interaction protocol is an extension of Algorithm 1, but instead of using direct communication, the teacher will allow the student to run some number of trajectories $m$ from a *fixed start state* and then decide whether to show a trace or not.

We would like to show that the length ($m$) of each observation phase can be polynomially bounded and the system as a whole can still maintain a good TI bound. We show below that such an $m$ exists and is related to the PAC-MDP bound for a portion of the environment we call the *zone of tractable exploration* (ZTE). The ZTE (inspired by the zone of proximal development [11]) is the area of an MDP that an agent with background knowledge $B$ and model learners $L_T$ and $L_R$ can act in with a polynomial number of suboptimal steps as judged only within that area. Combining the ZTE, $B$, $L_T$ and $L_R$ induces a learning sub-problem where the agent must learn to act as well as possible without the teacher's help.

**Remark 1.** *If the learning agent is KWIK-MBP and the evaluation phase has length $m = A_1 + A_2$ where $A_1$ is the PAC-MDP bound for the ZTE and $A_2$ is the number of trials all starting from $s_0$ needed to estimate $V_A(s_0)$ ($\hat{V}_A$) within accuracy $\epsilon/k$ for $k \geq 4$, and the teacher only steps in when $\hat{V}_A < V_T - \frac{(k-1)}{k}\epsilon$, the resulting interaction will have a TI bound equivalent to the earlier one, although the student needs to wait $m$ trials to get a trace from the teacher.*

$A_1$ trials are necessary because the agent may need to explore all the $\perp$ or optimistic mistakes within the ZTE, and each episode might contain only one of the $A_1$ suboptimal steps. Since each trajectory with a fixed policy results in an i.i.d. sample with mean $V_A$, $A_2$ can be polynomially bounded using a Chernoff bound [12]. Note we require here that $k \geq 4$ (a stricter requirement than earlier). This is because we have errors of $||V_A - \hat{V}_A|| \leq \epsilon/k$ and $||U_A - V_A|| \leq \epsilon/k$, so $\hat{V}_A$ needs to be at least $3\epsilon/k$ below $V_T$ to ensure $U_T < V_T - \epsilon/k$, and therefore traces are helpful. But $\hat{V}_A$ may also overestimate $V_A$, leading to an extra $\epsilon/k$ slack term, and hence $k \geq 4$.

# 6 Related Work and Conclusions

Our teaching protocol extends early apprenticeship learning work for linear MDPs [1], which showed a polynomial number of upfront traces followed by greedy (not explicitly exploring) trajectories could achieve good behavior. Our protocol is similar to a recent "practice/critique" interaction [13] where a teacher observed an agent and then labeled individual actions as "good" or "bad", but the teacher did not provide demonstrations in that work. Our setting differs from inverse reinforcement learning [4, 5] because our student can act better than the teacher, does not know the dynamics, and observes rewards. Studies have also been done on humans providing shaping rewards as feedback to agents rather than our demonstration technique [14, 15].

Some works have taken a heuristic approach to mixing autonomous learning and teacher-provided trajectories. This has been done in robot reinforcement learning domains [16] and for bootstrapping classifiers [17]. Many such approaches give all the teacher data at the beginning, while our teaching protocol has the teacher only step in selectively, and our theoretical results ensure the teacher will only step in when its advice will have a significant effect.

We have shown how to use an extension of the KWIK-MB [6] (now KWIK-MBP) framework as the basis for model-based RL agents in the apprenticeship paradigm. These agents have a "mixed" interpretation of their learned models that admits a degree of autonomous exploration. Furthermore, introducing a communication channel from the student to the teacher and having the teacher only give traces when $V_T$ is significantly better than $U_A$ guarantees the teacher will only provide demonstrations that attempt to teach concepts the agent could not tractably learn on its own, which has clear benefits when demonstrations are far more costly than exploration steps.

### Acknowledgments

We thank Michael Littman and Lihong Li for discussions and DARPA-27001328 for funding.

# References

[1] Pieter Abbeel and Andrew Y. Ng. Exploration and apprenticeship learning in reinforcement learning. In *ICML*, 2005.

[2] Richard S. Sutton and Andrew G. Barto. *Reinforcement Learning: An Introduction*. MIT Press, Cambridge, MA, March 1998.

[3] Thomas J. Walsh, Kaushik Subramanian, Michael L. Littman, and Carlos Diuk. Generalizing apprenticeship learning across hypothesis classes. In *ICML*, 2010.

[4] Pieter Abbeel and Andrew Y. Ng. Apprenticeship learning via inverse reinforcement learning. In *ICML*, 2004.

[5] Nathan Ratliff, David Silver, and J. Bagnell. Learning to search: Functional gradient techniques for imitation learning. *Autonomous Robots*, 27:25–53, 2009.

[6] Amin Sayedi, Morteza Zadimoghaddam, and Avrim Blum. Trading off mistakes and don't-know predictions. In *NIPS*, 2010.

[7] Lihong Li, Michael L. Littman, Thomas J. Walsh, and Alexander L. Strehl. Knows what it knows: A framework for self-aware learning. *Machine Learning*, 82(3):399–443, 2011.

[8] Alexander L. Strehl, Lihong Li, and Michael L. Littman. Reinforcement learning in finite MDPs: PAC analysis. *Journal of Machine Learning Research*, 10:2413–2444, 2009.

[9] Nick Littlestone. Learning quickly when irrelevant attributes abound. *Machine Learning*, 2:285–318, 1988.

[10] Dana Angluin. Queries and concept learning. *Machine Learning*, 2(4):319–342, 1988.

[11] Lev Vygotsky. Interaction between learning and development. In *Mind In Society*. Harvard University Press, Cambridge, MA, 1978.

[12] Michael J. Kearns, Yishay Mansour, and Andrew Y. Ng. Approximate planning in large pomdps via reusable trajectories. In *NIPS*, 1999.

[13] Kshitij Judah, Saikat Roy, Alan Fern, and Thomas G. Dietterich. Reinforcement learning via practice and critique advice. In *AAAI*, 2010.

[14] W. Bradley Knox and Peter Stone. Combining manual feedback with subsequent mdp reward signals for reinforcement learning. In *AAMAS*, 2010.

[15] Andrea Lockerd Thomaz and Cynthia Breazeal. Teachable robots: Understanding human teaching behavior to build more effective robot learners. *Artificial Intelligence*, 172(6-7):716–737, 2008.

[16] William D. Smart and Leslie Pack Kaelbling. Effective reinforcement learning for mobile robots. In *ICRA*, 2002.

[17] Sonia Chernova and Manuela Veloso. Interactive policy learning through confidence-based autonomy. *Journal of Artificial Intelligence Research*, 34(1):1–25, 2009.

